# Enhancing Robustness of Graph Neural Networks on Social Media with Explainable Inverse Reinforcement Learning

**Yuefei Lyu[1], Chaozhuo Li[1]\*, Sihong Xie[2], Xi Zhang[1]\***

[1]Key Laboratory of Trustworthy Distributed Computing and Service (BUPT)
Ministry of Education, Beijing University of Posts and Telecommunications, China
[2]Artificial Intelligence Thrust,
The Hong Kong University of Science and Technology (Guangzhou), China

## Abstract

Adversarial attacks against graph neural networks (GNNs) through perturbations of the graph structure are increasingly common in social network tasks like rumor detection. Social media platforms capture diverse attack sequence samples through both machine and manual screening processes. Investigating effective ways to leverage these adversarial samples to enhance robustness is imperative. We improve the maximum entropy inverse reinforcement learning (IRL) method with the mixture-of-experts approach to address multi-source graph adversarial attacks. This method reconstructs the attack policy, integrating various attack models and providing feature-level explanations, subsequently generating additional adversarial samples to fortify the robustness of detection models. We develop precise sample guidance and a bidirectional update mechanism to reduce the deviation caused by imprecise feature representation and negative sampling within the large action space of social graphs, while also accelerating policy learning. We take rumor detector as an example targeted GNN model on real-world rumor datasets. By utilizing a small subset of samples generated by various graph adversarial attack methods, we reconstruct the attack policy, closely approximating the performance of the original attack method. We validate that samples generated by the learned policy enhance model robustness through adversarial training and data augmentation.

## 1  Introduction

Social media platforms such as Weibo and Twitter host complex relationship networks that exhibit a typical graph structure. Graph neural networks play a crucial role in analyzing social graphs, demonstrating significant efficacy in a range of social network tasks, including rumor detection [1, 2, 3, 4, 5], spam detection [6, 7] and stance detection [8, 9].

Extensive research [10, 11] has demonstrated that GNNs are vulnerable to adversarial attacks, allowing adversaries to manipulate downstream node classification outcomes by flipping a small number of edges or features within the graph. For instance, in rumor detection, rumor spreaders may manipulate the graph structure to evade detection by reposting messages and following users. Recent research efforts have shifted towards acquiring such attack samples [1, 12], which can be obtained through manual supervision or effective detection models. Analyzing these samples allows platforms to profile attackers, uncover their motives, and understand their attack patterns, thereby crucially enhancing the robustness of detectors to defend against similar attacks.

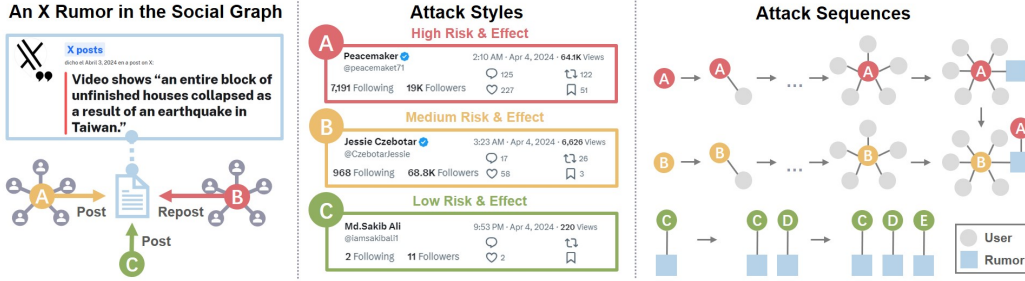

Figure 1: An X(Twitter) rumor published on Politifact. Three instances of the rumor's propagation on X are listed, which imply different attack styles and sequences: (1) User A, who has a notable number of followers, posted the rumor and gained considerable attention. The attack sequence involved purchasing followers and posting the rumor. (2) User B, with a large number of followers, retweeted the rumor originally posted by User A. This extends the last sequence to constitute a sequence of purchasing followers, posting, and retweeting. (3) User C, with almost no followers, copied the content posted by User A and published it again, forming a one-step attack sequence. Repeated similar attacks can be seen as a sequence.

Utilizing adversarial training, which augments model generalization by introducing perturbed samples into training data, is a common and effective approach [13, 14, 15, 16, 17]. The effectiveness of defense hinges crucially on the adversarial samples employed in adversarial training [18]. Current research on graph adversarial training primarily concentrates on devising effective attack methods to produce adversarial samples, thus fortifying the model's robustness against these attacks. However, within social networks, numerous attackers with diverse goals pose a challenge in developing attack methods that can simulate the wide array of real-world attacks, each characterized by different motives and styles, thus achieving comprehensive defense. Hence, we aim to reconstruct the attack policy to simulate multiple attackers as accurately as possible with the adversarial samples captured by social media platforms. It not only aids in gaining a more comprehensive understanding of the attackers but also facilitates the acquisition of additional samples for adversarial training.

A natural idea is to use the captured attack samples as training data for supervised learning, known as Behavior Cloning (BC) [19], which constitutes a form of simple imitation learning. However, within social networks, attack samples frequently demonstrate interdependencies, as attackers commonly execute multiple steps of graph perturbation behaviors to accomplish their ultimate objectives. For example, [1] proposed various camouflage behaviors aimed at deceiving rumor detectors. Rumor spreaders could combine several camouflage behaviors to evade from detection as shown in Figure 1. In sequential decision-making scenarios, BC encounters the challenge of compounding error [20, 21], as it leads to continual deviations from the observed sample distribution when faced with unknown states. Consequently, we can turn to inverse reinforcement learning (IRL) methods [22]. Diverging from reinforcement learning (RL) [23], IRL has access to some expert demonstrations while lacking knowledge of the reward function. Although the agent can interact with the environment, it does not obtain rewards; instead, it deduces the reward function concerning sample features from expert demonstrations and subsequently leverages reinforcement learning to uncover the optimal actor. IRL simulates adversarial attack policies based on observed data. Moreover, when employing linear reward functions and interpretable features, IRL offers feature-level post-hoc explanations [24], thus better aiding platform operators in understanding attack behaviors.

Reconstructing interpretable attack policies in social networks using inverse reinforcement learning poses two primary challenges. Firstly, expert demonstrations gathered from social media platforms originate from diverse attackers. Thus, it is imperative to develop a policy capable of simulating multiple experts as well as achieving similar attack performance. Secondly, while linear reward functions and interpretable features provide transparent interpretations, their application to represent graph structure data within the large action space of social graphs poses significant challenges. Similar sample features may correspond to entirely disparate ground true rewards [12], which make difficulties in inverse reinforcement learning.

Therefore, we propose MoE-BiEntIRL, an explainable bidirectional update maximum entropy inverse reinforcement learning method with the mindset of mixture-of-experts. It improves maximum entropy

inverse reinforcement learning (EntIRL) [25] to estimate interpretable linear reward function and recover the attack policy. It makes use of mixture-of-experts (MoE) model to cluster expert samples during the IRL process and learns optimal policies by leveraging the strengths of each expert, and provides feature-level explanations. To address the issue of suboptimal linear feature representation of graph structure data, we introduce precise sample guidance and bidirectional update mechanism to speed up the exploration of reinforcement learning and reward function learning.

**Contributions**. i) It studies a novel problem of reconstructing the attack policy with collected adversarial samples on social media. ii) Our approach enhances IRL techniques to handle the graph structured attack samples from diverse adversaries with large social graphs, while also offering interpretability. iii) On the real-world rumor datasets, we validate the policy reconstruction effectiveness of our method for multiple graph adversarial attack methods, and enhance the robustness of the GNNs rumor detector through data argumentation and adversarial training with additional samples generated by the reconstructed policies.

## 2 Background

**Reinforcement learning**. A Markov Decision Process (MDP) is defined as a tuple $(\mathcal{S}, \mathcal{A}, \mathcal{P}, r, \delta)$, where $\mathcal{S}$ is a set of states, $\mathcal{A}$ is a set of actions, $\mathcal{P}$ is the state transition probability function, $r$ is the reward function, and $\delta$ is the discount factor. The core objective in RL is to learn a policy $\pi : \mathcal{S} \to \mathcal{A}$ that maximizes the expected sum of discounted rewards: $V_\pi(s) = \mathbb{E}_\pi \left[ \sum_{t=0}^{\infty} \delta^t r(S_t, A_t) \mid S_0 = s \right]$, where $V_\pi(s)$ is the state-value function under policy $\pi$. RL algorithms typically learn the action-value function $Q_\pi(s, a)$, which represents the expected return of taking action $a$ in state $s$ and following policy $\pi$ thereafter. According to the Bellman Optimality Equation, the optimal action-value function is defined as $Q^*(s, a) = \mathbb{E}\left[ r(S_t, A_t) + \delta \max_{a'} Q^*(S_{t+1}, a') \mid S_t = s, A_t = a \right]$. The agent interacts with the environment to collect experiences, and then updates its policy or value functions to improve decision-making over time.

**Inverse reinforcement learning**. The core idea of IRL is to assume that the observed behavior is optimal with respect to some unknown reward function. The task is then to recover this reward function such that the learned policy is (near-)optimal under the recovered rewards. A classical formulation is Maximum Entropy IRL (EntIRL) [25]. Given a set of expert demonstrations $\mathcal{D}$, it seeks to recover the reward function with the principle of maximum entropy along the trajectory $\tau = \{s_0, a_0, ..., s_T\}$. Thus, the objective of EntIRL is defined as:

$$
\begin{aligned}
\max \sum_{\tau \in \mathcal{D}} -p(\tau) \log p(\tau) \\
\text{s.t.} \sum_{\tau \in \mathcal{D}} p(\tau) f_\tau = \tilde{f}, \sum_{\tau \in \mathcal{D}} p(\tau) = 1,
\end{aligned}
\tag{1}
$$

where $p(\tau)$ is the probability distribution of the trajectory, $f_\tau$ is the trajectory feature, and $\tilde{f}$ is the expert feature expectation. EntIRL assumes that $p(\tau) \propto e^{R_\theta(\tau)}$, where $R_\theta(\tau) = \sum_t r_\theta(s_t, a_t)$ is the cumulative reward with the reward function parameter $\theta$. With the limitation of feature matching, the maximum likelihood method naturally aligns with the maximum entropy principle [26]. Thus, the loss function for EntIRL is the likelihood as: $L(\theta) = \sum_{\tau \in \mathcal{D}} \log p(\tau|\theta)$. The locally optimal example like [27, 28] is considered here. It segments the trajectory as state-action pairs. Denoting the action $a$ from state $s$ along the trajectory $\tau$, the previous assumption becomes $p(a|s) \propto e^{Q^*(s,a)}$ [25]. With a discounting factor $\delta = 0$, the action probability is proportional to the exponential of the rewards encountered along $\tau$:

$$
p(a|s) = \frac{1}{Z} \exp(r_\theta(s, a)),
\tag{2}
$$

where $Z$ is the partition factor. The EntIRL loss function then becomes

$$
L(\theta) = \sum_{a \in \mathcal{A}_s} \log p(a|s),
\tag{3}
$$

where $\mathcal{A}_s$ is the action space under the state $s$. The linear reward function $r_\theta(s, a) = \theta^\top f(s, a)$ is adopted here with feature extraction method $f$. Then, the reward function is employed to train the learner policy $\pi_L$.

**Mixture-of-experts**. MoE is an ensemble model that consists of a gating network $\alpha$ and $K$ expert networks $\{p_1, p_2, ..., p_K\}$. Each expert is used to learn and store knowledge from different fields, and

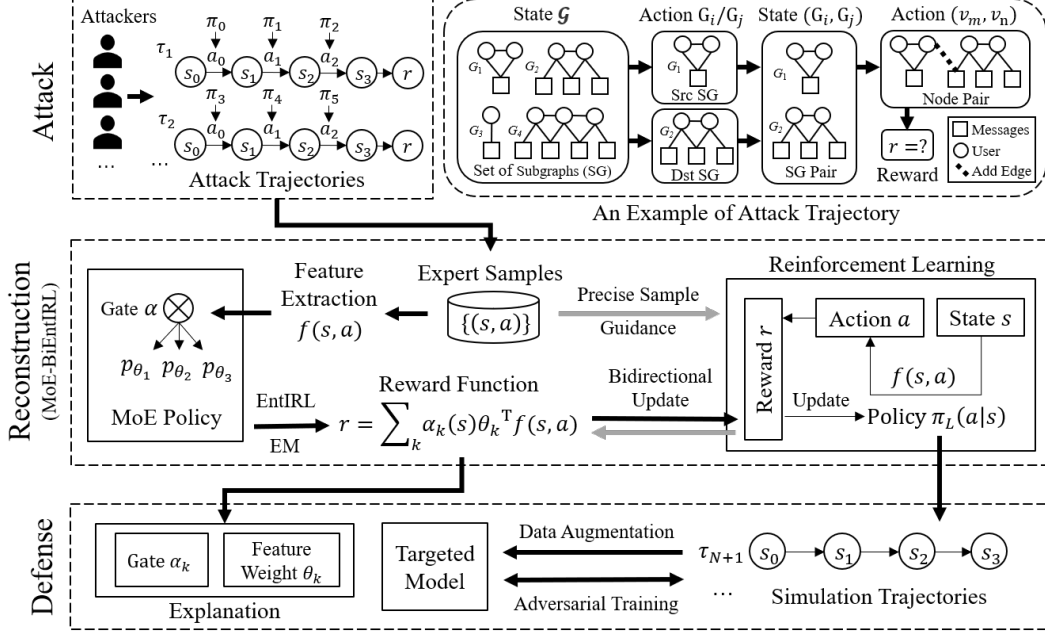

Figure 2: The framework of our proposal. There are three stages: attack, reconstruction and defense. An example of attack trajectory in social networks is shown at the top right.

the gating network determines the expert network used for the inference based on the input. For each input, the gating network dynamically selects expert networks for activation, which can be indicated as $p(y|x) = \sum_k \alpha(x)p_k(y|x)$.

**Graph adversarial attack on social networks**. The social network graph is denoted as $\mathcal{G}=(\mathcal{V},\mathcal{E})$. The node set $\mathcal{V}$ consists of the nodes representing messages, users and comments. The edge set $\mathcal{E}$ consists of pairs $(v_i, v_j)$, where $v_i, v_j \in \mathcal{V}$. Each edge or potential edge can be mapped to its relation type with the fucntion $\psi : \{(v_i, v_j)\} \to \mathcal{L}$. The relation type set $\mathcal{L}$ includes user-message, user-user and message-comment. The communities in social network are represented by a set of connected components $\{G_1, ..., G_m\}$ in $\mathcal{G}$, which are termed the *subgraph* in the sequel.

We focus on node classification task utilizing a GNN model, denoted as $g$. Each node $v_i$ in $\mathcal{G}$ is associated with a corresponding node label $y_i \in \mathcal{Y}$. Our setting is transductive, where the test nodes are observed during training without their labels. Within social networks, attackers aim to deceive the trained classifier $g$ through evasion attacks. We assume that the attackers can only observe a limited number of nodes and manipulate a subset of edges. The observable node set is denoted by $\mathcal{V}' \subset \mathcal{V}$. Here the modifiable edge set is defined as $\mathcal{E}' = \{(v_i, v_j) \mid v_i \in \mathcal{V}', v_j \in \mathcal{V}', \psi((v_i, v_j)) \in \mathcal{L}\}$, enabling the manipulation between observable nodes. Additionally, there is a target set $\mathcal{O} \subset \mathcal{V}$, allowing attackers to conduct global or targeted attacks within the controllable range by specifying the target set. The attacker modifies the graph $\mathcal{G}$ into $\tilde{\mathcal{G}}$. The objective of attackers is to maximize the cost function $L_A = \sum_{v_i \in \mathcal{O}} L(g(v_i), y_i)$ in $\tilde{\mathcal{G}}$, where $L(\cdot, \cdot)$ is the loss between two input values.

## 3 Method

### 3.1 Framework

As shown in Figure 2, there are three stages in our situation: attack, reconstruction, and defense. Initially, the attacker on social media perturbs the social network with a sequence of manipulations, constituting an MDP trajectory. Multiple attackers generate $N$ trajectories, denoted as $\{\tau_1, \tau_2, ..., \tau_N\}$, where each $\tau_j = (s_j^{(0)}, a_j^{(0)}, s_j^{(1)}, ..., a_j^{(T-1)}, s_j^{(T)})$. With the variety of attack styles and purposes, our proposal assumes distinct policies $\pi \in \{\pi_0, \pi_1, ...\}$ decide the action at each step. To expedite learning in large social network graphs, it decomposes edge flipping into three steps: source subgraph

selection, destination subgraph selection, and node pair selection. The final reward of the trajectory is $r$, which remains unknown to the targeted social media platform. Each state-action pair at time $t$ could be seen as an observed expert sample, with feature extraction method $f$ employed to represent these pairs. Given the features of samples, the MoE policy is introduced to enhance the EntIRL method, deducing the reward function with EM algorithm. Then, with the improvement of precise sample guidance and bidirectional update mechanism, the learner policy $\pi_L$ is optimized continuously. The policy $\pi_L$ could generate more trajectories $\{\tau_{N+1}, \tau_{N+2}, ...\}$ to simulate real attack samples. With data augmentation or adversarial training, it could enhance the robustness of the targeted model. Furthermore, by analyzing the reward function, it provides feature-level explanations of the attack samples.

## 3.2 Mixture-of-experts maximum entropy inverse reinforcement learning

### 3.2.1 Mixture-of-experts policy

With $N$ observed trajectories $\{\tau_1, \tau_2, ..., \tau_N\}$ and $\tau_j = (s_j^{(0)}, a_j^{(0)}, s_j^{(1)}, ..., a_j^{(T-1)}, s_j^{(T)})$, we assume that the trajectory is generated by the following model

$$p(\tau|\theta) = p(s^{(0)}) \times \prod_{t=0}^{T-1} p(a^{(t)}|s^{(t)})p(s^{(t+1)}|s^{(t)}, a^{(t)}), \tag{4}$$

and the policy is the mixture-of-experts model as

$$p(a^{(t)}|s^{(t)}, \theta) = \sum_{k=1}^{K} \alpha_k(s^{(t)}, \varphi)p(a^{(t)}|s^{(t)}, \theta_k), \tag{5}$$

where $\alpha_k(s^{(t)}, \varphi)$ is the gate function parametered by $\varphi$, and $\sum_{k=1}^{K} \alpha_k(s^{(t)}, \varphi) = 1$ with given $t$. There are $K$ experts with parameters $\theta = (\theta_1, ..., \theta_K)$ and each component $p(a^{(t)}|s^{(t)}, \theta_k)$ represents an expert. With Eq. (2) and $r_\theta(s, a) = \theta^\top f(s, a)$, we formulate the $k$-th expert at time $t$:

$$p(a^{(t)}|s^{(t)}, \theta_k) = \frac{\exp(\theta_k^\top f(s^{(t)}, a^{(t)}))}{\sum_{a \in \mathcal{A}_{s,t}} \exp(\theta_k^\top f(s^{(t)}, a))}, \tag{6}$$

where $\mathcal{A}_{s,t}$ is the action space under the state $s^{(t)}$ of the expert sample, and the denominator could be estimated with sampling.

### 3.2.2 EM algorithm

Each state-action pair is an observed sample. The expert to produce the $t$-th state-action pair in the observed trajectory $\tau_j$ is unknown. The latent variable $\gamma_{jkt} = 1$ if the $t$-th state-action pair of the trajectory $\tau_j$ is decided by the $k$-th expert, otherwise $\gamma_{jkt} = 0$. The complete data include observed trajectory $\tau_j$ and unobserved $\gamma_{jkt}$ with $j = 1, 2, .., N$. The likelihood function of complete data is

$$P(\tau, \gamma|\theta) = \prod_{j=1}^{N} P(\tau_j, \gamma_{j,1,0}, \gamma_{j,2,0}, ..., \gamma_{jKT})$$

$$= \prod_{j=1}^{N} \left[ p(s_j^{(0)}) \times \prod_{t=0}^{T-1} p(s_j^{(t+1)}|s_j^{(t)}, a_j^{(t)}) \right] \times \prod_{j=1}^{N} \prod_{t=0}^{T-1} \prod_{k=1}^{K} \left[ \alpha_k(s_j^{(t)})p(a_j^{(t)}|s_j^{(t)}, \theta_k) \right]^{\gamma_{jkt}}. \tag{7}$$

Then parameters $\theta$ are estimated by EM algorithm [29]:

**E-Step**: Given the observed data $(s_j^{(t)}, a_j^{(t)})$ and current parameters $\theta^{(i)}$, it computes the Q function as

$$\mathcal{Q}(\theta, \theta^{(i)}) = \mathbb{E}\left[ \log P(\tau, \gamma|\theta)|a_j^{(t)}, s_j^{(t)}, \theta^{(i)} \right]$$

$$= \sum_{t=0}^{T-1} \sum_{k=1}^{K} \sum_{j=1}^{N} \left( \hat{\gamma}_{jkt} \log \alpha_k(s_j^{(t)}) + \hat{\gamma}_{jkt} \log p(a_j^{(t)}|s_j^{(t)}, \theta_k) \right), \tag{8}$$

where $\hat{\gamma}_{jkt} = \mathbb{E}[\gamma_{jkt}]$ is the responsibility for the observed sample as

$$\hat{\gamma}_{jkt} = P(\gamma_{jkt} = 1|a_j^{(t)}, s_j^{(t)}, \theta^{(i)}) = \frac{\alpha_k(s_j^{(t)})p\left(a_j^{(t)}|s_j^{(t)}, \theta_k^{(i)}\right)}{\sum_{k=1}^{K} \alpha_k(s_j^{(t)})p(a_j^{(t)}|s_j^{(t)}, \theta_k^{(i)})}. \tag{9}$$

**M-Step**: The goal is updating $\theta$ with

$$\theta^{(i+1)} = \arg\max_{\theta} Q(\theta, \theta^{(i)}). \tag{10}$$

Then the gate function loss and the expert loss for the $k$-th expert are respectively

$$L_{gate}(\varphi) = \sum_{t=0}^{T-1} \sum_{k=1}^{K} \sum_{j=1}^{N} \hat{\gamma}_{jkt} \log \alpha_k(s_j^{(t)}), \tag{11}$$

$$L_{ex}(\theta_k) = \sum_{t=0}^{T-1} \sum_{j=1}^{N} \hat{\gamma}_{jkt} \log p(a_j^{(t)}|s_j^{(t)}, \theta_k). \tag{12}$$

The expert loss shares a conceptual basis with the EntIRL loss as Eq. (3), thus the EM algorithm can be employed to solve the IRL problem. With Eq. (6), the gradient of the normalized expert loss is

$$\nabla L_{ex}(\theta_k) = \tilde{f}_k - \frac{1}{NT} \sum_{t=0}^{T-1} \sum_{j=1}^{N} \hat{\gamma}_{jkt} \sum_{a \in \mathcal{A}_{s_j,t}} p(a|s_j^{(t)}, \theta_k) f(s_j^{(t)}, a). \tag{13}$$

where $\mathcal{A}_{s_j,t}$ is the action space for state $s_j^{(t)}$, and $\tilde{f}_k$ is the feature expectations for the $k$-th expert:

$$\tilde{f}_k = \frac{1}{NT} \sum_{t=0}^{T-1} \sum_{j=1}^{N} \hat{\gamma}_{jkt} f(s_j^{(t)}, a_j^{(t)}). \tag{14}$$

It updates each $\theta_k$ with gradient ascent and action space sampling. With learned $\theta_k$, the reward can be estimated by

$$r_\theta(s, a) = \sum_{k=1}^{K} \alpha_k(s)\theta_k^\top f(s, a). \tag{15}$$

### 3.3 Improvement mechanism

**Precise sample guidance**. In inverse reinforcement learning, the objective is to obtain an accurate reward function from expert demonstrations, enabling the learner policy to approximate the expert's behavior. Adversaries utilize specific feature extraction methods $f'$, to obtain embeddings for each attack behavior. However, in the context of social media, the feature extraction method $f'$ employed by the attack model is unknown. The surrogate feature extraction method $f$ is employed to simulate the input of the attack model, replacing $f'$ with $f$. This imprecise feature representation implies a sensitive mapping from features to rewards, where minor discrepancies could lead to significantly different attack rewards [12, 30].

According to the reconstruction process along the black arrows in Figure 2, the learner policy $\pi_L$ cannot directly observe the raw expert samples. The information delivering of expert samples involves feature extraction, reward function estimation and RL policy update. This process necessitates both computations and sampling. Deviations in feature representation can accumulate, resulting in a reward function that inadequately guides policy learning. Consequently, we introduce expert structural perturbations directly during the policy learning process, allowing the learner policy to replicate expert sample actions rather than relying on the imprecise features. Specifically, in the initial stage of policy learning, we enforce the learner policy to execute expert actions at a predetermined frequency and assign maximum reward values to the trajectory.

**Bidirectional update mechanism**. While the precise guidance mechanism speeds up exploration in reinforcement learning, it does not facilitate reward function learning in inverse reinforcement learning. Reward function learning relies solely on the EM algorithm with expert demonstrations as input, as indicated by Eq. (13). Both feature representation deviations and action space sampling also impact reward function learning. By executing expert demonstrations through precise guidance and assuming they yield maximum rewards, we can incorporate this information into the parameter updates of the reward function. Specifically, during the precise sample guidance phase, we perform inverse updates with the loss

$$L_{inv}(\theta_k) = \alpha_k(s)L(\hat{r}, \theta_k^\top f(s, a)). \tag{16}$$

where $(s, a)$ is the expert sample selected for enforcement and $\hat{r}$ is the maximum historical reward value. This process provides feedback opposite to the output of the reward function, ensuring synchronized learning of the learner policy and the reward function. During normal reinforcement learning phases without precise sample guidance, we update $\theta_k$ according to Eq. (13).

## 3.4 Threat model and defense on social networks

**Algorithm 1:** MoE-BiEntIRL

**Input:** Expert demonstration set $\mathcal{D}$, number of expert demonstration $N$, number of experts $K$, length of trajectories $T$, number of episodes $E$, feature extraction function $f$, gate function $\alpha$, reward function parameters $\theta = (\theta_1, \ldots, \theta_K)$, learner policy $\pi_L$, negative sample set $\mathcal{D}'$, responsibility matrix $\gamma \in \mathbb{R}^{N \times T \times K}$, inverse update episode set $\Lambda$

**Output:** Learner policy $\pi_L$

1 **for** $e = 1, 2, \ldots, E$ **do**
2    $s = \text{env\_reset}()$;
3    **for** $t = 1, 2, \ldots, T$ **do**
4      **if** $e \in \Lambda$ **then**
5        $s', a = \text{precise\_sample\_guidance}(\mathcal{D})$;
6        $r = \text{max\_reward}()$;
7      **else**
8        $s', a = \text{env\_step}(s, \pi_L)$;
9        $r = \text{obtain\_reward}(\alpha, f(s, a), \theta)$ as Eq. (15);
10      $\pi_L = \text{update\_policy}(s, a, s', r)$;
11      $s = s'$;
12    $\gamma = \text{calculate\_responsibility}(\alpha, \mathcal{D}, \mathcal{D}', \theta)$ as Eq. (9);
13    **for** $k = 1, 2, \ldots, K$ **do**
14      **if** $e \in \Lambda$ **then**
15        $\theta_k = \text{inverse\_update}(\alpha, \mathcal{D}, \theta_k)$ as Eq. (16);
16      **else**
17        $\theta_k = \text{gradient\_ascent}(\gamma, \mathcal{D}, \mathcal{D}', \theta_k)$ as Eq. (13);
18    $\alpha = \text{gradient\_ascent}(\gamma)$ with the loss as Eq. (11);

With precise sample guidance and bidirectional update mechanism, we improve the mixture-of-experts EntIRL to MoE-BiEntIRL as the threat model to reconstruct the attack policy. The overall algorithm is as shown in Algorithm 1 and the time complexity analysis is shown in Appendix D. There are some details of MoE-BiEntIRL for the node classification task on social media.

**Hierarchical reinforcement learning.** Inspired by [12], the hierarchical RL is employed and improved here, which includes three layers as illustrated in Figure 2: the source subgraph, the destination subgraph, and the node pair. In the source or destination subgraph layer, the state is the graph $\mathcal{G}$ and the action is a subgraph $G_i$ or $G_j$ at time $t$. In the node pair layer, the state and the action are the subgraph pair $(G_i, G_j)$ and a node pair $(v_m, v_n)$, respectively. Each layer is governed by a policy. The state-action pairs correspond to the selection of source subgraph $G_i$, destination subgraph $G_j$ or node pair $(v_m, v_n)$. We employ a linear action-value function to learn the policy with LinUCB algorithm [31], and it could be replaced with other RL methods.

**Interpretable features.** For feature extraction method $f$, comprehensible features can be employed to represent attack actions within the graph, as suggested by [32], facilitating the derivation of an interpretable reward function. The graph and node features designed in [12] are utilized here, focusing specifically on targeting rumor detectors. The details are shown in Appendix C.

**Sampling.** In the process of IRL, two sampling procedures are involved, as indicated in Eq. (6) and Eq. (13). These procedures necessitate the sampling of state-action pairs to represent the action space. Here similarity negative sampling is adopted under specified assumptions. This method selects state-action pairs $(s, a)$ based on the following criteria: i) Samples with a high similarity to expert samples are prioritized, under the condition that the cosine similarity $cos(f(s, a), f(s', a')) < \mu$, where $(s', a') \in \mathcal{D}$ represents an expert sample. ii) In cases where $(s, a)$ represents the selection of a subgraph, there must be target nodes in the source subgraph $G_i$ and controllable nodes in the destination subgraph $G_j$.

**Defense with adversarial samples.** With the trained learner policy parameterized by $\omega$, we can generate additional samples to attack the targeted model $g$ parameterized by $\sigma$. Denoting the ground truth for node $v_i$ as $y_i$, the predictions on clean and perturbed graphs are represented by $y_i'$ and $\tilde{y}_i'$, respectively. Robustness of the targeted model can be improved through offline data augmentation or online adversarial training. The overall loss is given by

$$L_D = \sum_i L_\sigma(y_i, y_i') + \beta \sum_i L_{\sigma,\omega}(y_i, \tilde{y}_i'). \tag{17}$$

In data augmentation, $\sigma$ is updated while $\omega$ remains fixed. In adversarial training, the process can be viewed as a minimax game: $\min_\sigma \max_\omega L_D(\sigma, \omega)$. The attacker adjusts $\omega$ to maximize the loss, while the defender alternately updates $\sigma$ to minimize the loss.

## 4 Experiments

**Dataset**. Our focus is on the rumor detection task, for which we conduct experiments on two real-world datasets: Weibo [33] and Pheme (event *ferguson* in [34]). These datasets contain both rumors and non-rumors, along with associated user, reposting, and comment data. Specifically, due to the limited number of following relationships among users, we connect edges between user nodes as described in [12], leveraging the potential user communities inferred from their posting messages. Dataset statistics are provided in Table 1. The datasets are split into training and testing sets using a 7:3 ratio. We reconstruct the policy during the training phase and implement the defense during the testing phase. In the training set, 20% of the authors and their posting

Table 1: Dataset statistics.

|  | Weibo | Pheme |
|---|---|---|
| Nodes | 10,280 | 2,708 |
| Edges | 16,412 | 4,401 |
| Rumors | 1,538 | 284 |
| Non-rumors | 1,849 | 859 |
| Users | 2,440 | 1,008 |
| Comments | 4,453 | 557 |

messages are designated as controllable nodes, while all nodes in the testing set are considered controllable. All rumor nodes in the controllable set are regarded as target nodes, forming the set $\mathcal{O}$. Attackers are only permitted to add edges between controllable users and messages.

**Target model**. The rumor detection model is a 2-layer GCN [35]. The detection accuracy is shown in Table 3. The hidden layer dimension is 64. Message node embedding is represented using a fixed text embedding layer during attacks. It is trained over 60 and 120 epochs for Weibo and Pheme, respectively, and employs the Adam optimizer with a learning rate of 0.0001.

**Metric**. Attack performance is measured using

$$\Delta L_A = L_A(0) - L_A(T). \tag{18}$$

Here, the attack loss $L_A = \sum_{v_i \in \mathcal{O}}(g(v_i) - y_i)$ represents the total loss between the rumor probability and the ground truth for the target nodes in $\mathcal{O}$. $L_A(0)$ and $L_A(T)$ denote the attack loss in the clean graph and after $T$-step attacks, respectively. $T$ serves as the horizon for RL and defines the budget for graph adversarial attacks. Specifically, it limits the modification of $T$ edges when attacking.

**Attack method**. The expert samples are collected through four graph adversarial attack methods, categorized based on their attack cost. Rule-based *PageRank* and black-box *GC-RWCS* are considered low-cost attacks. High-cost attacks include *PR-BCD* with a white-box setting and *AdRumor-RL* with complete feature knowledge.

- *PageRank*. This method establishes connections between users and messages with high influence, measured using the PageRank algorithm. Inspired by [1] and [12], it selectively links rumors with normal users or non-rumors with malicious users.
- *GC-RWCS* [11]. Utilizing a black-box node selection strategy, this method employs a greedy procedure to determine node importance scores. Here it connects messages with high importance scores to influential users, utilizing the same influence measure and limited edge types as *PageRank*.
- *PR-BCD* [36]. This is a sparsity-aware first-order optimization graph adversarial attack method targeting GNNs in a white-box setting. It proposes the surrogate loss for global attacks.
- *AdRumor-RL* [12]. This hierarchical contextual bandit attack framework targets GCN-based rumor detectors using interpretable features. It is an RL-based method to produce the serialized attack trajectories with a black-box setting.

**Attack performance evaluation of policy reconstruction.** We reconstruct policies using expert samples generated by the aforementioned attack methods via MoE-BiEntIRL. The IRL process consists of 1000 episodes. The precise sample guidance and bidirectional update mechanism is applied every two episodes during the first 500 episodes. For each attack method, we select the top one to three attack trajectories based on performance as expert samples. The number of experts, $K$, is estimated using DBSCAN and is typically adjusted to range from 1 to 25, often being less than 10. The gate function is pre-trained with the labels generated by Gaussian Mixture Model (GMM) clustering. To prevent cluster collapse, we adopt the weighted sum of one-hot labels and gate function outputs as the MoE gate. Each expert sample is augmented with 100 negative samples with a sampling upper bound $\mu = 0.8$. All experiments are conducted using GTX 2080Ti (11GB) GPUs.

We compare MoE-BiEntIRL with two classical IRL methods: (i) *Apprenticeship Learning* [37], which is based on maximum margin and faces an ill-posed problem introducing ambiguity. (ii) *EntIRL*

Table 2: The attack performance of policy reconstruction evaluated using the average value of the last 100 episodes in the metric $\Delta L_A$ as shown in Eq. (18). A higher $\Delta L_A$ means better performance. The rows and columns correspond to the IRL methods and attack models used to generate expert samples, respectively. The row Expert is the average performance of the expert samples. In the column of Mixture, we display the performance of the policy reconstructed with expert samples from all low or high cost attack methods.

| | | High-Cost Attack | | | Low-Cost Attack | | |
|---|---|---|---|---|---|---|---|
| | | *PRBCD* | *AdRumor* | Mixture | *PageRank* | *GC-RWCS* | Mixture |
| Weibo T=5 | Expert | 4.865 | 4.877 | - | 3.000 | 3.000 | - |
| | *Apprenticeship* | 1.275 | 0.788 | 0.704 | 0.850 | 0.763 | 1.071 |
| | *EntIRL* | 4.650 | 4.770 | 4.550 | **5.000** | 4.950 | 4.950 |
| | *MoE-BiEntIRL* | **4.989** | **4.990** | **4.929** | 4.860 | 4.900 | 4.900 |
| Weibo T=20 | Expert | 19.521 | 19.854 | - | 5.449 | 5.160 | - |
| | *Apprenticeship* | 1.142 | 3.066 | 3.945 | 0.030 | 0.040 | 0.020 |
| | *EntIRL* | 19.030 | 19.749 | 19.199 | 19.830 | **20.000** | **20.000** |
| | *MoE-BiEntIRL* | **19.876** | **19.936** | **19.979** | **19.970** | 19.700 | 18.749 |
| Pheme T=5 | Expert | 4.804 | 5.947 | - | 2.991 | 3.990 | - |
| | *Apprenticeship* | 1.788 | 3.387 | 2.619 | 0.000 | 0.000 | 0.000 |
| | *EntIRL* | 0.000 | 0.018 | 0.010 | 0.000 | 0.062 | 0.000 |
| | *MoE-BiEntIRL* | **2.205** | **4.965** | **4.277** | **1.488** | **2.105** | **1.549** |

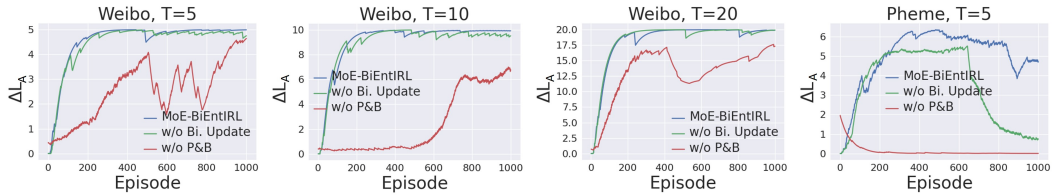

Figure 3: The smoothed curves of the ablation experiments for precise sample guidance and bidirectional update mechanism when recovering the policy of *AdRumor-RL*. The terms *w/o Bi.Update* and *w/o P&B* denote the removal of the bidirectional update mechanism and the removal of both improvement modules, respectively.

[25], which applies the maximum entropy theory to IRL to alleviate ambiguity. The performance is shown in Table 2. Our findings indicate that: i) MoE-BiEntIRL outperforms the expert policy on Weibo. ii) Our method excels in reconstructing policies for high-cost attacks, while for simple low-cost attacks, *EntIRL* often outperforms due to *Occam's Razor*; however, our method also achieves comparable results. iii) Despite suboptimal effects on Pheme compared to experts, other IRL methods struggle to learn the policy, showcasing the difficulty of policy recovery. Furthermore, we validate the effectiveness of the precise sample guidance and bidirectional update mechanism, as depicted in Figure 3, particularly advantageous in challenging policy reconstruction scenarios.

**The improvement of robustness with generated samples**. We assess the efficacy of samples produced by MoE-BiEntIRL in enhancing robustness by subjecting the target model to attacks from *PageRank*, *GC-RWCS*, and *PR-BCD*. Evaluation is conducted under various conditions: no defense, data augmentation with expert samples (EDA), data augmentation with generated samples (DA), and adversarial training (AT). The trade-off parameter $\beta$ in Eq. (17) is set to 8. Results are presented in Table 3. Simply using all expert samples for data augmentation does not yield effective defensive results. In contrast, employing additional samples generated by MoE-BiEntIRL exhibit the highest or second-highest improvement in robustness while maintaining accuracy in the clean graph.

**Case study for interpretability**. Our analysis focuses on the expert samples generated by *AdRumor-RL*, which relies on a linear action-value function, with the policy parameter providing insight into feature importance. In our approach, we determine the feature importance of the sample $(s, a)$ through the calculation $\sum \theta_k^\top \alpha_k(s)$. Table 4 illustrates the top eight important features elucidated

Table 3: The test accuracy decline (%) of the GCN rumor detector with $T = 5$ on Weibo dataset. The first row displays the attack method. The first column is the way to enhance the robustness with adversarial samples. The second column is the method to generate the samples. The column under w/o Att. reflects test accuracy without attacks, while results under other columns reflect accuracy decline. The second row (w/o Def.) shows the accuracy (decline) without any defense method. **Boldfaced font** and $*$ mean the best performance and the runner-up among all methods respectively.

|  |  | w/o Att. | *PageRank* | *GC-RWCS* | *PR-BCD* |
|---|---|---|---|---|---|
|  | w/o Def. | 70.4031 | -0.4042 | -0.4406 | -0.1966 |
| EDA | *PageRank* | 70.5998 | **-0.1821** | **-0.2440** | **0.0000** |
|  | *GC-RWCS* | 70.7965 | -0.4043 | -0.4407 | -0.1967 |
|  | *PR-BCD* | 70.3048 | -0.2185 | **-0.2440** | **0.0000** |
|  | *AdRumor-RL* | 70.7965 | $*$-0.2076 | **-0.2440** | **0.0000** |
|  | All above | 70.7965 | -0.2805 | -0.3424 | -0.0984 |
| DA | *PageRank* | 70.6981 | -0.5025 | -0.5390 | -0.2950 |
|  | *GC-RWCS* | 70.5015 | -0.3059 | -0.2440 | 0.0000 |
|  | *PR-BCD* | 70.4031 | **-0.1092** | **-0.1456** | **0.0984** |
|  | *AdRumor-RL* | 70.6981 | -0.3059 | -0.3423 | -0.0983 |
|  | *MoE-BiEntIRL* | 70.6981 | **-0.1092** | **-0.1456** | **0.0984** |
| AT | *PageRank* | 71.0914 | **-0.2075** | **-0.2440** | **0.0000** |
|  | *GC-RWCS* | 70.2065 | -0.4042 | -0.4407 | -0.1967 |
|  | *PR-BCD* | 70.4031 | -0.3059 | -0.3423 | -0.0983 |
|  | *AdRumor-RL* | 70.6981 | -0.3059 | -0.3423 | -0.0983 |
|  | *MoE-BiEntIRL* | 72.0747 | $*$-0.2731 | $*$-0.2589 | **0.0000** |

Table 4: The top-8 important features for subgraph selection with $T = 5$ on Weibo dataset, reflected by *AdRumor-RL* expert samples and the learned reward function, respectively. The overlapping features are marked with the gray background. The features are described in Appendix C.

| Expert sample | Reward function |
|---|---|
| The Source Subgraph | |
| max potential | rumor review |
| rumor review | n nodes |
| avg degree | n edges |
| n nodes | max degree |
| author ratio | max potential |
| n edges | message ratio |
| avg potential | max rumor inf |
| author inf min | review ratio |
| The Destination Subgraph | |
| avg potential | user inf min |
| rumor ratio | avg user inf |
| min user inf | avg author inf |
| min author inf | min author inf |
| avg user inf | max user inf |
| avg author inf | max author inf |
| max author inf | max nonrumor inf |
| max user inf | avg nonrumor inf |

by both *AdRumor-RL* and our method. The reward function enables the capture of the majority of important features.

## 5 Related Work

**Graph adversarial attack and defense**. Graph adversarial attacks include poisoning [38] and evasion attacks [39], as well as global [40] and targeted attacks [10], spanning both white-box and black-box approaches [10, 11]. Adversarial samples are utilized in numerous studies to train robust GNNs through adversarial training techniques [18, 13, 14]. As for rumor detection, some studies explore graph adversarial attacks on social networks, as evidenced by [1, 4, 12, 41, 42, 43, 44].

**Inverse reinforcement learning**. It includes maximum margin-based approaches like apprenticeship learning [37] and probability model-based methods such as EntIRL [25] and REIRL [45]. Regarding explainable IRL, [46] explores potential clustering factors in demonstrations, offering expert-level explanations. [24] quantifies the importance of different goals in ICU hypotension management with linear reward function. Additionally, [47] also explores the combination of MoE and EntIRL based on decision trees.

## 6 Conclusion

We propose MoE-BiEntIRL, a threat model to recover the graph adversarial attack policy against GNN model on social media. It utilizes the multi-source graph structured attack trajectories to learn a generalized policy based on IRL techniques and MoE mindset, and provides feature-level explanations. The precise sample guidance and bidirectional update mechanism are designed to deal with the deviation caused by feature representation and negative sampling. Leveraging samples produced by the reconstructed policy, it could enhance the robustness of the target model. The broader impact and the limitation of our work are shown in Appendix A and B, respectively.

## Acknowledgement

Yuefei Lyu, Chaozhuo Li and Xi Zhang were supported by the Natural Science Foundation of China (No. 62372057). Sihong Xie was supported in part by the National Key R&D Program of China (Grant No. 2023YFF0725001), the Guangzhou-HKUST(GZ) Joint Funding Program (Grant No. 2023A03J0008), and Education Bureau of Guangzhou Municipality. This material is based upon work supported by the National Science Foundation under Grant Number 2008155 & 1931042.

## Footnotes

\*Corresponding authors: lichaozhuo@bupt.edu.cn, zhangx@bupt.edu.cn.

## References

[1] Xiaoyu Yang, Yuefei Lyu, Tian Tian, Yifei Liu, Yudong Liu, and Xi Zhang. Rumor detection on social media with graph structured adversarial learning. In *Proceedings of the twenty-ninth international conference on international joint conferences on artificial intelligence*, pages 1417–1423, 2021.

[2] Tian Bian, Xi Xiao, Tingyang Xu, Peilin Zhao, Wenbing Huang, Yu Rong, and Junzhou Huang. Rumor detection on social media with bi-directional graph convolutional networks. In *Proceedings of the AAAI conference on artificial intelligence*, volume 34, pages 549–556, 2020.

[3] Van-Hoang Nguyen, Kazunari Sugiyama, Preslav Nakov, and Min-Yen Kan. Fang: Leveraging social context for fake news detection using graph representation. In *Proceedings of the 29th ACM international conference on information & knowledge management*, pages 1165–1174, 2020.

[4] Tiening Sun, Zhong Qian, Sujun Dong, Peifeng Li, and Qiaoming Zhu. Rumor detection on social media with graph adversarial contrastive learning. In Frédérique Laforest, Raphaël Troncy, Elena Simperl, Deepak Agarwal, Aristides Gionis, Ivan Herman, and Lionel Médini, editors, *WWW '22: The ACM Web Conference 2022, Virtual Event, Lyon, France, April 25 - 29, 2022*, pages 2789–2797. ACM, 2022.

[5] Yi-Ju Lu and Cheng-Te Li. Gcan: Graph-aware co-attention networks for explainable fake news detection on social media. *arXiv preprint arXiv:2004.11648*, 2020.

[6] Binghui Wang, Neil Zhenqiang Gong, and Hao Fu. Gang: Detecting fraudulent users in online social networks via guilt-by-association on directed graphs. In *2017 IEEE International Conference on Data Mining (ICDM)*, pages 465–474. IEEE, 2017.

[7] Ao Li, Zhou Qin, Runshi Liu, Yiqun Yang, and Dong Li. Spam review detection with graph convolutional networks. In *Proceedings of the 28th ACM International conference on information and knowledge management*, pages 2703–2711, 2019.

[8] John Pougué-Biyong, Akshay Gupta, Aria Haghighi, and Ahmed El-Kishky. Learning stance embeddings from signed social graphs. In Tat-Seng Chua, Hady W. Lauw, Luo Si, Evimaria Terzi, and Panayiotis Tsaparas, editors, *Proceedings of the Sixteenth ACM International Conference on Web Search and Data Mining, WSDM 2023, Singapore, 27 February 2023 - 3 March 2023*, pages 177–185. ACM, 2023.

[9] Chen Li, Hao Peng, Jianxin Li, Lichao Sun, Lingjuan Lyu, Lihong Wang, Philip S. Yu, and Lifang He. Joint stance and rumor detection in hierarchical heterogeneous graph. *IEEE Trans. Neural Networks Learn. Syst.*, 33(6):2530–2542, 2022.

[10] Hanjun Dai, Hui Li, Tian Tian, Xin Huang, Lin Wang, Jun Zhu, and Le Song. Adversarial attack on graph structured data. In *International conference on machine learning*, pages 1115–1124. PMLR, 2018.

[11] Jiaqi Ma, Shuangrui Ding, and Qiaozhu Mei. Towards more practical adversarial attacks on graph neural networks. *Advances in neural information processing systems*, 33:4756–4766, 2020.

[12] Yuefei Lyu, Xiaoyu Yang, Jiaxin Liu, Sihong Xie, Philip S. Yu, and Xi Zhang. Interpretable and effective reinforcement learning for attacking against graph-based rumor detection. In *International Joint Conference on Neural Networks, IJCNN 2023, Gold Coast, Australia, June 18-23, 2023*, pages 1–9. IEEE, 2023.

[13] Kaidi Xu, Hongge Chen, Sijia Liu, Pin-Yu Chen, Tsui-Wei Weng, Mingyi Hong, and Xue Lin. Topology attack and defense for graph neural networks: An optimization perspective. *arXiv preprint arXiv:1906.04214*, 2019.

[14] Fuli Feng, Xiangnan He, Jie Tang, and Tat-Seng Chua. Graph adversarial training: Dynamically regularizing based on graph structure. *IEEE Transactions on Knowledge and Data Engineering*, 33(6):2493–2504, 2019.

[15] Zhijie Deng, Yinpeng Dong, and Jun Zhu. Batch virtual adversarial training for graph convolutional networks. *AI Open*, 4:73–79, 2023.

[16] Kaidi Xu, Sijia Liu, Pin-Yu Chen, Mengshu Sun, Caiwen Ding, Bhavya Kailkhura, and Xue Lin. Towards an efficient and general framework of robust training for graph neural networks. In *2020 IEEE International Conference on Acoustics, Speech and Signal Processing, ICASSP 2020, Barcelona, Spain, May 4-8, 2020*, pages 8479–8483. IEEE, 2020.

[17] Jintang Li, Jiaying Peng, Liang Chen, Zibin Zheng, Tingting Liang, and Qing Ling. Spectral adversarial training for robust graph neural network. *IEEE Trans. Knowl. Data Eng.*, 35(9):9240–9253, 2023.

[18] Lukas Gosch, Simon Geisler, Daniel Sturm, Bertrand Charpentier, Daniel Zügner, and Stephan Günnemann. Adversarial training for graph neural networks: Pitfalls, solutions, and new directions. *Advances in Neural Information Processing Systems*, 36, 2024.

[19] Dean A Pomerleau. Alvinn: An autonomous land vehicle in a neural network. *Advances in neural information processing systems*, 1, 1988.

[20] Stéphane Ross, Geoffrey Gordon, and Drew Bagnell. A reduction of imitation learning and structured prediction to no-regret online learning. In *Proceedings of the fourteenth international conference on artificial intelligence and statistics*, pages 627–635. JMLR Workshop and Conference Proceedings, 2011.

[21] Stephen Tu, Alexander Robey, Tingnan Zhang, and Nikolai Matni. On the sample complexity of stability constrained imitation learning. In *Learning for Dynamics and Control Conference*, pages 180–191. PMLR, 2022.

[22] Andrew Y Ng, Stuart Russell, et al. Algorithms for inverse reinforcement learning. In *Icml*, volume 1, page 2, 2000.

[23] Richard S Sutton and Andrew G Barto. *Reinforcement learning: An introduction*. MIT press, 2018.

[24] Srivatsan Srinivasan and Finale Doshi-Velez. Interpretable batch irl to extract clinician goals in icu hypotension management. *AMIA Summits on Translational Science Proceedings*, 2020:636, 2020.

[25] Brian D Ziebart, Andrew L Maas, J Andrew Bagnell, Anind K Dey, et al. Maximum entropy inverse reinforcement learning. In *Aaai*, volume 8, pages 1433–1438. Chicago, IL, USA, 2008.

[26] E. T. Jaynes. Information theory and statistical mechanics. *Phys. Rev.*, 106:620–630, May 1957.

[27] Sergey Levine and Vladlen Koltun. Continuous inverse optimal control with locally optimal examples. In *Proceedings of the 29th International Conference on Machine Learning, ICML 2012, Edinburgh, Scotland, UK, June 26 - July 1, 2012*. icml.cc / Omnipress, 2012.

[28] Nathan D. Ratliff, Brian D. Ziebart, Kevin M. Peterson, J. Andrew Bagnell, Martial Hebert, Anind K. Dey, and Siddhartha S. Srinivasa. Inverse optimal heuristic control for imitation learning. In David A. Van Dyk and Max Welling, editors, *Proceedings of the Twelfth International Conference on Artificial Intelligence and Statistics, AISTATS 2009, Clearwater Beach, Florida, USA, April 16-18, 2009*, volume 5 of *JMLR Proceedings*, pages 424–431. JMLR.org, 2009.

[29] Todd K Moon. The expectation-maximization algorithm. *IEEE Signal processing magazine*, 13(6):47–60, 1996.

[30] Sili Huang, Yanchao Sun, Jifeng Hu, Siyuan Guo, Hechang Chen, Yi Chang, Lichao Sun, and Bo Yang. Learning generalizable agents via saliency-guided features decorrelation. In Alice Oh, Tristan Naumann, Amir Globerson, Kate Saenko, Moritz Hardt, and Sergey Levine, editors, *Advances in Neural Information Processing Systems 36: Annual Conference on Neural Information Processing Systems 2023, NeurIPS 2023, New Orleans, LA, USA, December 10 - 16, 2023*, 2023.

[31] Lihong Li, Wei Chu, John Langford, and Robert E. Schapire. A contextual-bandit approach to personalized news article recommendation. In *WWW*, pages 661–670, 2010.

[32] Junhao Zhu, Yalu Shan, Jinhuan Wang, Shanqing Yu, Guanrong Chen, and Qi Xuan. Deepinsight: Interpretability assisting detection of adversarial samples on graphs. *CoRR*, abs/2106.09501, 2021.

[33] Changhe Song, Cheng Yang, Huimin Chen, Cunchao Tu, Zhiyuan Liu, and Maosong Sun. Ced: Credible early detection of social media rumors. *TKDE*, 33(8):3035–3047, 2021.

[34] Arkaitz Zubiaga, Maria Liakata, and Rob Procter. Learning reporting dynamics during breaking news for rumour detection in social media. *CoRR*, abs/1610.07363, 2016.

[35] Thomas N. Kipf and Max Welling. Semi-supervised classification with graph convolutional networks. In *ICLR*, 2017.

[36] Simon Geisler, Tobias Schmidt, Hakan Sirin, Daniel Zügner, Aleksandar Bojchevski, and Stephan Günnemann. Robustness of graph neural networks at scale. In Marc'Aurelio Ranzato, Alina Beygelzimer, Yann N. Dauphin, Percy Liang, and Jennifer Wortman Vaughan, editors, *Advances in Neural Information Processing Systems 34: Annual Conference on Neural Information Processing Systems 2021, NeurIPS 2021, December 6-14, 2021, virtual*, pages 7637–7649, 2021.

[37] Pieter Abbeel and Andrew Y. Ng. Apprenticeship learning via inverse reinforcement learning. In Carla E. Brodley, editor, *Machine Learning, Proceedings of the Twenty-first International Conference (ICML 2004), Banff, Alberta, Canada, July 4-8, 2004*, volume 69 of *ACM International Conference Proceeding Series*. ACM, 2004.

[38] Aleksandar Bojchevski and Stephan Günnemann. Adversarial attacks on node embeddings via graph poisoning. In *ICML*, volume 97 of *Proceedings of Machine Learning Research*, pages 695–704, 2019.

[39] Huijun Wu, Chen Wang, Yuriy Tyshetskiy, Andrew Docherty, Kai Lu, and Liming Zhu. Adversarial examples for graph data: Deep insights into attack and defense. In *IJCAI*, pages 4816–4823, 7 2019.

[40] Daniel Zügner and Stephan Günnemann. Adversarial attacks on graph neural networks via meta learning. In *ICLR*, 2019.

[41] Hao Yan, Chaozhuo Li, Ruosong Long, Chao Yan, Jianan Zhao, Wenwen Zhuang, Jun Yin, Peiyan Zhang, Weihao Han, Hao Sun, et al. A comprehensive study on text-attributed graphs: Benchmarking and rethinking. In *Thirty-seventh Conference on Neural Information Processing Systems Datasets and Benchmarks Track*, 2023.

[42] Chaozhuo Li, Senzhang Wang, Dejian Yang, Zhoujun Li, Yang Yang, Xiaoming Zhang, and Jianshe Zhou. Ppne: property preserving network embedding. In *Database Systems for Advanced Applications: 22nd International Conference, DASFAA 2017, Suzhou, China, March 27-30, 2017, Proceedings, Part I 22*, pages 163–179. Springer, 2017.

[43] Jianan Zhao, Meng Qu, Chaozhuo Li, Hao Yan, Qian Liu, Rui Li, Xing Xie, and Jian Tang. Learning on large-scale text-attributed graphs via variational inference. *arXiv preprint arXiv:2210.14709*, 2022.

[44] Peiyan Zhang, Jiayan Guo, Chaozhuo Li, Yueqi Xie, Jae Boum Kim, Yan Zhang, Xing Xie, Haohan Wang, and Sunghun Kim. Efficiently leveraging multi-level user intent for session-based recommendation via atten-mixer network. In *Proceedings of the Sixteenth ACM International Conference on Web Search and Data Mining*, pages 168–176, 2023.

[45] Abdeslam Boularias, Jens Kober, and Jan Peters. Relative entropy inverse reinforcement learning. In Geoffrey J. Gordon, David B. Dunson, and Miroslav Dudík, editors, *Proceedings of the Fourteenth International Conference on Artificial Intelligence and Statistics, AISTATS 2011, Fort Lauderdale, USA, April 11-13, 2011*, volume 15 of *JMLR Proceedings*, pages 182–189. JMLR.org, 2011.

[46] Yunzhu Li, Jiaming Song, and Stefano Ermon. Infogail: Interpretable imitation learning from visual demonstrations. In Isabelle Guyon, Ulrike von Luxburg, Samy Bengio, Hanna M. Wallach, Rob Fergus, S. V. N. Vishwanathan, and Roman Garnett, editors, *Advances in Neural Information Processing Systems 30: Annual Conference on Neural Information Processing Systems 2017, December 4-9, 2017, Long Beach, CA, USA*, pages 3812–3822, 2017.

[47] Marko Vasic, Andrija Petrovic, Kaiyuan Wang, Mladen Nikolic, Rishabh Singh, and Sarfraz Khurshid. Moët: Mixture of expert trees and its application to verifiable reinforcement learning. *Neural Networks*, 151:34–47, 2022.

[48] Siyuan Guo, Lixin Zou, Hechang Chen, Bohao Qu, Haotian Chi, Philip S. Yu, and Yi Chang. Sample efficient offline-to-online reinforcement learning. *IEEE Trans. Knowl. Data Eng.*, 36(3):1299–1310, 2024.

## A Broader impact

Our method, rooted in social media analysis, harnesses platform data to bolster model robustness against attacks. Attackers face substantial data capture expenses and may find our method unsuitable for their particular objectives. In conclusion, we posit that the benefits of our approach surpass its drawbacks concerning social impact.

## B Limitation

IRL commonly suffers from suboptimal expert samples, leading to noise in policy learning. The diverse nature of attackers on social media results in varied sample effects, and platforms may erroneously capture normal samples. In this work, we categorize expert samples based on attack cost to mitigate excessive variance in their true reward within an IRL context. Nonetheless, real-world implementation of such grouping may encounter challenges in accuracy and feasibility. Future research will focus on enhancing IRL resilience to noise and developing effective pre-classification methods. In addition, it faces the distribution shift problem [48] due to the different distribution among the adversaries training data, the negative sampling data and the RL action space.

## C Interpretable features

The interpretable features are designed by [12]. Detailed descriptions are provided in Table 5 and Table 6. Features not within the range [0,1] are normalized using Min-Max normalization. With regard to the features present in Table 4, as long as the features have the same suffix, they are considered to capture the same important feature.

Table 5: The subgraph level features. # refers to the number of. $a : b$ means the ratio of $a$ to $b$. * can be replaced with avg/max/min here, which means average/maximum/minimum.

| Name | Description | Name | Description |
|---|---|---|---|
| **Structural features** | | | |
| n nodes | # nodes. | clustering coeff. | The global clustering coefficient. |
| n edges | # edges. | * degree | The avg/max/min node degree. |
| **Social features** | | | |
| message rat. | # message nodes : # nodes. | author rat. | # author nodes : # nodes. |
| rumor rat. | # rumor nodes : # message nodes. | bad author rat. | # bad author nodes : # author nodes. |
| retweeter rat. | # retweeter nodes : # nodes. | review rat. | # comment nodes : # nodes. |
| rumor retweet | # retweeter nodes who connect to rumor nodes : # retweeter nodes. | rumor review | # comment nodes who connect to rumor nodes : # comment nodes. |
| **Influence features** | | | |
| * author inf | The avg/max/min influence of | * rumor inf | The avg/max/min influence of |
| * user inf | author/user nodes. | * nonrumor inf | (non-)rumor nodes. |
| **Attack potential features** | | | |
| * potential | The avg/max/min probability of target rumors. | attack degree | # added edges in the previous steps : horizon $T$. |
| **Ranking help message features** | | | |
| * rhm suspicious | The avg/max/min probability of non-target rumors. | - | - |

## D The time complexity

**The time complexity of MoE-BiEntIRL**. There are three principal phases of MoE-BiEntIRL and we indicate the corresponding lines of Algorithm 1 as follows.

Table 6: The node level features. # refers to the number of. $a : b$ means the ratio of $a$ to $b$. * can be replaced with avg/max/min here, which means average/maximum/minimum.

| Name | Description | Name | Description |
|------|-------------|------|-------------|
| **Structural features** | | | |
| degree | The degree of the node. | ego n edges | # edges in the ego network. |
| **Social features** | | | |
| good bad | 0 if the node is good author or non-rumor, 1 if the node is bad author or rumor. | ego rumor rat. ego bu rat. | # rumor nodes : # nodes in the ego network. # bad author nodes : # nodes in the ego network. |
| node type | The one-hot vector to indicate the node type (rumor, non-rumor, good author, bad author). | ego review rat. | # comment nodes : # nodes in the ego network. |
| **Influence features** | | | |
| ego user inf ego message inf | The average influence of the user/ message nodes in the ego network. | node inf | The influence of the node. |
| **Attack potential features** | | | |
| * node potential | The avg/max/min probability of target rumors within 1-hop insides. | n node attack degree | # added edges that connect to the node in the previous steps : horizon $T$. |
| * neighbor suspicious | The avg/max/min probability of target rumors within the node 3-hop insides. | n targets distance | The average distance from the node to the target rumors within the node 3-hop insides. |
| n targets | # target rumors within the node $k$-hop insides. | | |
| **Ranking help message features** | | | |
| * rhm suspicious | The avg/max/min probability of non-target rumors within the node 3-hop insides. | - | - |

- Phase I - interaction (line 8): Our proposal conducts $T$-step attacks and interacts with the environment. Each attack step requires feature updates for the involved subgraphs and nodes, including a forward pass on the target model. For the $L$-layer GCN as the target model, the time complexity of the forward pass is approximately $O(LN_{\mathrm{e}}d + LN_{\mathrm{n}}d^2)$, where $N_{\mathrm{n}}$ and $N_{\mathrm{e}}$ denote the numbers of nodes and edges in the input graph, respectively, and $d$ is the feature dimension.

- Phase II - reward acquisition (line 12, 17 and 18): The reward is estimated with the IRL module, including responsibility calculation, gradient ascent, and gate function updates. The time complexity for an episode is $O(NTKd(S + K))$, where $N$ is the number of expert trajectories, $T$ is the trajectory length, $K$ is the number of experts, $d$ is the feature dimension, and $S$ is the number of negative samples.

- Phase III - policy update (line 10): In our work, we employ the LinUCB algorithm, characterized by a policy update time complexity of $O(d^2)$.

Overall, the total time complexity for an episode is approximately $O(TLN_{\mathrm{e}}d+TLN_{\mathrm{n}}d^2+NTKd(S+K)+d^2)$. Here the precise sample guidance (line 5 and 6) and bidirectional update mechanism (line 15) are not considered, which would reduce the time complexity in practice.

**The time complexity analysis of baselines**. The time complexities of our proposal and baselines are summarized in Table 7. The complexities of the interaction phase and policy update phase in the baseline models are identical to those in the proposed model, differing only in the reward acquisition phase. The time complexities of the reward acquisition phase within the baselines are

- *Apprenticeship Learning*: $O(NTd)$;
- *EntIRL*: $O(NTdS)$.

**Summary**. Based on Table 7, the time complexities of our proposal and the baseline methods primarily diverge during the reward acquisition phase of IRL. Our approach introduces $K$ experts

Table 7: This table outlines the time complexity and runtime of the MoE-BiEntIRL model proposed herein, alongside two baseline models. Time complexity is delineated across three principal phases: (I) interaction, (II) reward acquisition, and (III) policy update. The reported total running time denotes the average duration of a single episode. Additionally, specific attention is given to the runtime of reward acquisition phase for clarity and comprehensive evaluation. The runtime of experiments on Weibo and Pheme is displayed with $T$=5 and $N$=3.

| | Time complexity | | | Total / Phase II Runtime | |
|---|---|---|---|---|---|
| | Phase I | Phase II | Phase III | Weibo | Pheme |
| *Apprenticeship* | $O(TLN_\text{e}d + TLN_\text{n}d^2)$ | $O(NTd)$ | $O(d^2)$ | 4.52 / 0.02 | 0.55 / 0.02 |
| *EntIRL* | $O(TLN_\text{e}d + TLN_\text{n}d^2)$ | $O(NTdS)$ | $O(d^2)$ | 5.01 / 0.09 | 0.67 / 0.07 |
| *MoE-BiEntIRL* | $O(TLN_\text{e}d + TLN_\text{n}d^2)$ | $O(NTKd(S + K))$ | $O(d^2)$ | 5.31 / 0.36 | 0.81 / 0.24 |

to manage multi-source attack trajectories with diverse motivations, thereby increasing the time complexity. Notably, the complexity associated with reward acquisition is independent of the input graph size and hinges solely on the predefined hyperparameter $K$, typically ranging from 1 to 10, rendering the increased complexity manageable. Moreover, the predominant computational effort across all models is concentrated in the interaction phase, which further mitigates the impact of introducing multiple experts. Table 7 details the runtimes of complete episodes and the phase II. Despite the extended runtime of the MoE-BiEntIRL approach in the reward acquisition phrase, it exerts negligible influence on the total runtime. Both the analysis of time complexity and experimental findings emphasize that the actual runtime is largely influenced by the interaction phase. Therefore, the additional complexity introduced by employing MoE within the reward acquisition phase remains manageable.

